# Convergence and Energy Landscape for Cheeger Cut Clustering

**Xavier Bresson**
City University of Hong Kong
Hong Kong
xbresson@cityu.edu.hk

**Thomas Laurent**
University of California, Riversize
Riverside, CA 92521
laurent@math.ucr.edu

**David Uminsky**
University of San Francisco
San Francisco, CA 94117
duminsky@usfca.edu

**James H. von Brecht**
University of California, Los Angeles
Los Angeles, CA 90095
jub@math.ucla.edu

## Abstract

This paper provides both theoretical and algorithmic results for the $\ell_1$-relaxation of the Cheeger cut problem. The $\ell_2$-relaxation, known as spectral clustering, only loosely relates to the Cheeger cut; however, it is convex and leads to a simple optimization problem. The $\ell_1$-relaxation, in contrast, is non-convex but is provably equivalent to the original problem. The $\ell_1$-relaxation therefore trades convexity for exactness, yielding improved clustering results at the cost of a more challenging optimization. The first challenge is understanding convergence of algorithms. This paper provides the first complete proof of convergence for algorithms that minimize the $\ell_1$-relaxation. The second challenge entails comprehending the $\ell_1$-energy landscape, i.e. the set of possible points to which an algorithm might converge. We show that $\ell_1$-algorithms can get trapped in local minima that are not globally optimal and we provide a classification theorem to interpret these local minima. This classification gives meaning to these suboptimal solutions and helps to explain, in terms of graph structure, when the $\ell_1$-relaxation provides the solution of the original Cheeger cut problem.

## 1 Introduction

Partitioning data points into sensible groups is a fundamental problem in machine learning. Given a set of data points $V = \{x_1, \cdots, x_n\}$ and similarity weights $\{w_{i,j}\}_{1 \leq i,j \leq n}$, we consider the balance Cheeger cut problem [4]:

$$\text{Minimize} \quad \mathcal{C}(S) = \frac{\sum_{x_i \in S} \sum_{x_j \in S^c} w_{i,j}}{\min(|S|, |S^c|)} \qquad \text{over all subsets } S \subsetneq V. \tag{1}$$

Here $|S|$ denotes the number of data points in $S$ and $S^c$ is the complementary set of $S$ in $V$. While this problem is NP-hard, it has the following *exact continuous $\ell_1$-relaxation*:

$$\text{Minimize} \quad E(f) = \frac{\frac{1}{2} \sum_{i,j} w_{i,j} |f_i - f_j|}{\sum_i |f_i - \text{med}(f)|} \qquad \text{over all non-constant functions } f : V \to \mathbb{R}. \tag{2}$$

Here $\text{med}(f)$ denotes the median of $f \in \mathbb{R}^n$ and $f_i \equiv f(x_i)$. Recently, various algorithms have been proposed [12, 6, 7, 1, 9, 5] to minimize $\ell_1$-relaxations of the Cheeger cut (1) and of other related problems. Typically these $\ell_1$-algorithms provide excellent unsupervised clustering results

and improve upon the standard $\ell_2$ (spectral clustering) method [10, 13] in terms of both Cheeger energy and classification error. However, complete theoretical guarantees of convergence for such algorithms do not exist. This paper provides the first proofs of convergence for $\ell_1$-algorithms that attempt to minimize (2).

In this work we consider two algorithms for minimizing (2). We present a new steepest descent (SD) algorithm and also consider a slight modification of the inverse power method (IPM) from [6]. We provide convergence results for both algorithms and also analyze the energy landscape. Specifically, we give a complete classification of local minima. This understanding of the energy landscape provides intuition for when and how the algorithms get trapped in local minima. Our numerical experiments show that the two algorithms perform equally well with respect to the quality of the achieved cut. Both algorithms produce state of the art unsupervised clustering results. Finally, we remark that the SD algorithm has a better theoretical guarantee of convergence. This arises from the fact that the distance between two successive iterates necessarily converges to zero. In contrast, we cannot guarantee this holds for the IPM without further assumptions on the energy landscape. The simpler mathematical structure of the SD algorithm also provides better control of the energy descent.

Both algorithms take the form of a fixed point iteration $f^{k+1} \in \mathcal{A}(f^k)$, where $f \in \mathcal{A}(f)$ implies that $f$ is a critical point. To prove convergence towards a fix point typically requires three key ingredients: the first is monotonicity of $\mathcal{A}$, that is $E(z) \leq E(f)$ for all $z \in \mathcal{A}(f)$; the second is some estimate that guarantees the successive iterates remain in a compact domain on which $E$ is continuous; lastly, some type of continuity of the set-valued map $\mathcal{A}$ is required. For set valued maps, closedness provides the correct notion of continuity [8]. Monotonicity of the IPM algorithm was proven in [6]. This property alone is not enough to obtain convergence, and the closedness property proves the most challenging ingredient to establish for the algorithms we consider. Section 2 elucidates the form these properties take for the SD and IPM algorithms. In Section 3 we show that that if the iterates of either algorithm approach a neighborhood of a strict local minimum then both algorithms will converge to this minimum. We refer to this property as local convergence. When the energy is non-degenerate, section 4 extends this local convergence to global convergence toward critical points for the SD algorithm by using the additional structure afforded by the gradient flow. In Section 5 we develop an understanding of the energy landscape of the continuous relaxation problem. For non-convex problems an understanding of local minima is crucial. We therefore provide a complete classification of the local minima of (2) in terms of the combinatorial local minima of (1) by means of an explicit formula. As a consequence of this formula, the problem of finding local minima of the combinatorial problem is equivalent to finding local minima of the continuous relaxation. The last section is devoted to numerical experiments.

We now present the SD algorithm. Rewrite the Cheeger functional (2) as $E(f) = T(f)/B(f)$, where the numerator $T(f)$ is the total variation term and the denominator $B(f)$ is the balance term. If $T$ and $B$ were differentiable, a mixed explicit-implicit gradient flow of the energy would take the form $(f^{k+1} - f^k)/\tau^k = -(\nabla T(f^{k+1}) - E(f^k)\nabla B(f^k))/(B(f^k))$, where $\{\tau^k\}$ denotes a sequence of time steps. As $T$ and $B$ are not differentiable, particularly at the binary solutions of paramount interest, we must consider instead their subgradients

$$\partial T(f) := \{v \in \mathbb{R}^n : T(g) - T(f) \geq \langle v, g - f \rangle \; \forall g \in \mathbb{R}^n \}, \tag{3}$$
$$\partial_0 B(f) := \{v \in \mathbb{R}^n : B(g) - B(f) \geq \langle v, g - f \rangle \; \forall g \in \mathbb{R}^n \text{ and } \langle \mathbf{1}, v \rangle = 0 \}. \tag{4}$$

Here $\mathbf{1} \in \mathbb{R}^n$ denotes the constant vector of ones. Also note that if $f$ has zero median then $B(f) = ||f||_1$ and $\partial_0 B(f) = \{v \in \text{sign}(f), \text{s.t. mean}(v) = 0\}$. After an appropriate choice of time steps we arrive to the SD Algorithm summarized in table 1(on left), i.e. a non-smooth variation of steepest descent. A key property of the the SD algorithm's iterates is that $||f^{k+1} - f^k||_2 \to 0$. This property allows us to conclude global convergence of the SD algorithm in cases where we can not conclude convergence for the IPM algorithm. We also summarize the IPM algorithm from [6] in Table 1 (on right). Compared to the original algorithm from [6], we have added the extra step to project onto the sphere $\mathcal{S}^{n-1}$, that is $f^{k+1} = h^k/||h^k||_2$. While we do not think that this extra step is essential, it simplifies the proof of convergence.

The successive iterates of both algorithms belong to the space

$$\mathcal{S}_0^{n-1} := \{f \in \mathbb{R}^n : ||f||_2 = 1 \quad \text{and} \quad \text{med}(f) = 0\}. \tag{5}$$

Table 1: $\mathcal{A}_{\text{SD}}$ : SD Algorithm.                    $\mathcal{A}_{\text{IPM}}$ : Modifed IPM Algorithm [6].

| | |
|---|---|
| $f^0$ nonzero function with $\text{med}(f) = 0$. | $f^0$ nonzero function with $\text{med}(f) = 0$. |
| $c$ positive constant. | **while** $E(f^k) - E(f^{k+1}) \geq \text{TOL}$ **do** |
| **while** $E(f^k) - E(f^{k+1}) \geq \text{TOL}$ **do** | $\quad v^k \in \partial_0 B(f^k)$ |
| $\quad v^k \in \partial_0 B(f^k)$ | $\quad D^k = \min_{\lVert u \rVert_2 \leq 1} T(u) - E(f^k)\langle u, v^k \rangle$ |
| $\quad g^k = f^k + c\, v^k$ | $\quad g^k = \arg\min_{\lVert u \rVert_2 \leq 1} T(u) - E(f^k)\langle u, v^k \rangle$ if $D^k < 0$ |
| $\quad \hat{h}^k = \arg\min_{u \in \mathbb{R}^n} T(u) + \frac{E(f^k)}{2c}\lVert u - g^k \rVert_2^2$ | $\quad g^k = f^k$ if $D^k = 0$ |
| $\quad h^k = \hat{h}^k - \text{med}(\hat{h}^k)\mathbf{1}$ | $\quad h^k = g^k - \text{med}(g^k)\mathbf{1}$ |
| $\quad f^{k+1} = \frac{h^k}{\lVert h^k \rVert_2}$ | $\quad f^{k+1} = \frac{h^k}{\lVert h^k \rVert_2}$ |
| **end while** | **end while** |

As the successive iterates have zero median, $\partial_0 B(f^k)$ is never empty. For example, we can take $v^k \in \mathbb{R}^n$ so that $v^k(x_i) = 1$ if $f(x_i) > 0$, $v^k(x_i) = -1$ if $f(x_i) < 0$ and $v^k(x_i) = (n^- - n^+)/(n_0)$ if $f(x_i) = 0$ where $n^+$, $n^-$ and $n^0$ denote the cardinalities of the sets $\{x_i : f(x_i) > 0\}$, $\{x_i : f(x_i) > 0\}$ and $\{x_i : f(x_i) = 0\}$, respectively. Other possible choices also exist, so that $v^k$ is not uniquely defined. This idea, i.e. choosing an element from the subdifferential with mean zero, was introduced in [6] and proves indispensable when dealing with median zero functions. As $v^k$ is not uniquely defined in either algorithm, we must introduce the concepts of a *set-valued map* and a *closed map*, which is the proper notion of continuity in this context:

**Definition 1** (Set-valued Map, Closed Maps)**.** *Let $X$ and $Y$ be two subsets of $\mathbb{R}^n$. If for each $x \in X$ there is a corresponding set $F(x) \subset Y$ then $F$ is called a **set-valued map** from $X$ to $Y$. We denote this by $F : X \rightrightarrows Y$. The graph of $F$, denoted $Graph(F)$ is defined by*

$$Graph(F) = \{(x, y) \in \mathbb{R}^n \times \mathbb{R}^n : x \in X, \ y \in F(x)\}.$$

*A set-valued map $F$ is called **closed** if $Graph(F)$ is a closed subset of $\mathbb{R}^n \times \mathbb{R}^n$.*

With these notations in hand we can write $f^{k+1} \in \mathcal{A}_{\text{SD}}(f^k)$ (SD algorithm) and $f^{k+1} \in \mathcal{A}_{\text{IPM}}(f^k)$ (IPM algorithm) where $\mathcal{A}_{\text{SD}}, \mathcal{A}_{\text{IPM}} : \mathcal{S}_0^{n-1} \rightrightarrows \mathcal{S}_0^{n-1}$ are the appropriate set-valued maps. The notion of a closed map proves useful when analyzing the step $\hat{h}^k \in \mathcal{H}(f^k)$ in the SD algorithm. Particularly,

**Lemma 1** (Closedness of $\mathcal{H}(f)$)**.** *The following set-valued map $\mathcal{H} : \mathcal{S}_0^{n-1} \rightrightarrows \mathbb{R}^n$ is closed.*

$$\mathcal{H}(f) := \arg\min_u \left\{ T(u) + \frac{E(f)}{2c}\lVert u - (f + c\partial_0 B(f)) \rVert_2^2 \right\}$$

Currently, we can only show that lemma 1 holds at strict local minima for the analogous step, $g^k$, of the IPM algorithm. That lemma 1 holds without this further restriction on $f \in \mathcal{S}_0^{n-1}$ will allow us to demonstrate stronger global convergence results for the SD algorithm. Due to page limitations the supplementary material contains the proofs of all lemmas and theorems in this paper.

## 2 Properties of $\mathcal{A}_{\text{SD}}$ and $\mathcal{A}_{\text{IPM}}$

This section establishes the required properties of the of the set-valued maps $\mathcal{A}_{\text{SD}}$ and $\mathcal{A}_{\text{IPM}}$ mentioned in the introduction. In section 2.1 we first elucidate the monotonicity and compactness of $\mathcal{A}_{\text{SD}}$ and $\mathcal{A}_{\text{IPM}}$. Section 2.2 demonstrates that a local notion of closedness holds for each algorithm. This form of closedness suffices to show *local* convergence toward isolated local minima (c.f. Section 3). In particular, this more difficult and technical section is necessary as monotonicity alone does not guarantee this type of convergence.

### 2.1 Monotonicity and Compactness

We provide the monotonicity and compactness results for each algorithm in turn. Lemmas 2 and 3 establish monotonicity and compactness for $\mathcal{A}_{\text{SD}}$ while Lemmas 4 and 5 establish monotonicity and compactness for $\mathcal{A}_{\text{IPM}}$.

**Lemma 2** (Monotonicity of $\mathcal{A}_{\mathrm{SD}}$)**.** *Let* $f \in \mathcal{S}_0^{n-1}$ *and define* $v, g, \hat{h}$ *and* $h$ *according to the SD algorithm. Then neither* $\hat{h}$ *nor* $h$ *is a constant vector. Moreover, the energy inequality*

$$E(f) \geq E(h) + \frac{E(f)}{B(h)} \frac{\|\hat{h} - f\|_2^2}{c} \tag{6}$$

*holds. As a consequence, if* $z \in \mathcal{A}_{\mathrm{SD}}(f)$ *then* $E(z) = E(h) < E(f)$ *unless* $z = f$.

**Lemma 3** (Compactness of $\mathcal{A}_{\mathrm{SD}}$)**.** *Let* $f^0 \in \mathcal{S}_0^{n-1}$ *and define a sequence of iterates* $(g^k, \hat{h}^k, h^k, f^{k+1})$ *according to the SD algorithm. Then for any such sequence*

$$\|\hat{h}^k\|_2 \leq \|g^k\|_2, \quad 1 \leq \|g^k\|_2 \leq 1 + c\sqrt{n} \quad and \quad 0 < \|h^k\|_2 \leq (1 + \sqrt{n})\|\hat{h}^k\|_2. \tag{7}$$

*Moreover, we have*

$$\|\hat{h}^k - f^k\|_2 \to 0, \qquad \mathrm{med}(\hat{h}^k) \to 0, \qquad \|f^k - f^{k+1}\|_2 \to 0. \tag{8}$$

*Therefore* $\mathcal{S}_0^{n-1}$ *attracts the sequences* $\{\hat{h}^k\}$ *and* $\{h^k\}$.

By the monotonicity result of Hein and Bühler [6] we have

**Lemma 4** (Monotonicity of $\mathcal{A}_{\mathrm{IPM}}$)**.** *Let* $f \in \mathcal{S}_0^{n-1}$. *If* $z \in \mathcal{A}_{\mathrm{IPM}}(f)$ *then* $E(z) < E(f)$ *unless* $z = f$.

To prove convergence for $\mathcal{A}_{\mathrm{IPM}}$ using our techniques, we must also maintain control over the iterates after subtracting the median. This control is provided by the following lemma.

**Lemma 5** (Compactness of $\mathcal{A}_{\mathrm{IPM}}$)**.** *Let* $f \in \mathcal{S}_0^{n-1}$ *and define* $v, D, g$ *and* $h$ *according to the IPM.*

1. *The minimizer is unique when* $D < 0$*, i.e.* $g \in \mathcal{S}^{n-1}$ *is a single point.*

2. $1 \leq \|h\|_2 \leq 1 + \sqrt{n}$*. In particular,* $\mathcal{A}_{\mathrm{IPM}}(f)$ *is always well-defined for a given choice of* $v \in \partial_0 B(f)$.

## 2.2 Closedness Properties

The final ingredient to prove local convergence is some form of closedness. We require closedness of the set valued maps $\mathcal{A}$ at strict local minima of the energy. As the energy (2) is invariant under constant shifts and scalings, the usual notion of a strict local minimum on $\mathbb{R}^n$ does not apply. We must therefore remove the effects of these invariances when referring to a local minimum as strict. To this end, define the spherical and annular neighborhoods on $\mathcal{S}_0^{n-1}$ by

$$\mathcal{B}_\epsilon(f^\infty) := \{\|f - f^\infty\|_2 \leq \epsilon\} \cap \mathcal{S}_0^{n-1} \quad \mathcal{A}_{\delta,\epsilon}(f^\infty) := \{\delta \leq \|f - f^\infty\|_2 \leq \epsilon\} \cap \mathcal{S}_0^{n-1}.$$

With these in hand we introduce the proper definition of a strict local minimum.

**Definition 2** (Strict Local Minima)**.** *Let* $f^\infty \in \mathcal{S}_0^{n-1}$*. We say* $f^\infty$ *is a **strict local minimum** of the energy if there exists* $\epsilon > 0$ *so that* $f \in \mathcal{B}_\epsilon(f^\infty)$ *and* $f \neq f^\infty$ *imply* $E(f) > E(f^\infty)$.

This definition then allows us to formally define closedness at a strict local minimum in Definition 3. For the IPM algorithm this is the only form of closedness we are able to establish. Closedness at an arbitrary $f \in \mathcal{S}_0^{n-1}$ (c.f. lemma 1) does in fact hold for the SD algorithm. Once again, this fact manifests itself in the stronger global convergence results for the SD algorithm in section 4.

**Definition 3** (CLM/CSLM Mappings)**.** *Let* $\mathcal{A}(f) : \mathcal{S}_0^{n-1} \rightrightarrows \mathcal{S}_0^{n-1}$ *denote a set-valued mapping. We say* $\mathcal{A}(f)$ *is **closed at local minima** (CLM) if* $z^k \in \mathcal{A}(f^k)$ *and* $f^k \to f^\infty$ *imply* $z^k \to f^\infty$ *whenever* $f^\infty$ *is a local minimum of the energy. If* $z^k \to f^\infty$ *holds only when* $f^\infty$ *is a strict local minimum then we say* $\mathcal{A}(f)$ *is **closed at strict local minima** (CSLM).*

The CLM property for the SD algorithm, provided by lemma 6, follows as a straight forward consequence of lemma 1. The CSLM property for the IPM algorithm provided by lemma 7 requires the additional hypothesis that the local minimum is strict.

**Lemma 6** (CLM Property for $\mathcal{A}_{\mathrm{SD}}$)**.** *For* $f \in \mathcal{S}_0^{n-1}$ *define* $g, \hat{h}$ *and* $h$ *according to the SD algorithm. Then* $\mathcal{A}_{\mathrm{SD}}(f)$ *defines a CLM mapping.*

**Lemma 7** (CSLM Property for $\mathcal{A}_{\mathrm{IPM}}$)**.** *For* $f \in \mathcal{S}_0^{n-1}$ *define* $v, D, g, h$ *according to the IPM. Then* $\mathcal{A}_{\mathrm{IPM}}(f)$ *defines a CSLM mapping.*

# 3 Local Convergence of $\mathcal{A}_{\mathrm{SD}}$ and $\mathcal{A}_{\mathrm{IPM}}$ at Strict Local Minima

Due to the lack of convexity of the energy (2) , at best we can only hope to obtain convergence to a local minimum of the energy. An analogue of Lyapunov's method from differential equations allows us to show that such convergence does occur provided the iterates reach a neighborhood of an isolated local minimum. To apply the lemmas from section 2 we must assume that $f^\infty \in \mathcal{S}_0^{n-1}$ is a local minimum of the energy. We will assume further that $f^\infty$ is an isolated critical point of the energy according to the following definition.

**Definition 4** (Isolated Critical Points). *Let $f \in \mathcal{S}_0^{n-1}$. We say that $f$ is a **critical point** of the energy $E(f)$ if there exist $w \in \partial T(f)$ and $v \in \partial_0 B(f)$ so that $0 = w - E(f)v$. This generalizes the usual quotient rule $0 = \nabla T(f) - E(f)\nabla B(f)$. If there exists $\epsilon > 0$ so that $f$ is the only critical point in $\mathcal{B}_\epsilon(f^\infty)$ we say $f$ is an **isolated critical point** of the energy.*

Note that as any local minimum is a critical point of the energy, if $f^\infty$ is an isolated critical point and a local minimum then it is necessarily a strict local minimum. The CSLM property therefore applies.

Finally, to show convergence, the set-valued map $\mathcal{A}$ must possess one further property, i.e. the critical point property.

**Definition 5** (Critical Point Property). *Let $\mathcal{A}(f) : \mathcal{S}_0^{n-1} \rightrightarrows \mathcal{S}_0^{n-1}$ denote a set-valued mapping. We say that $\mathcal{A}(f)$ satisfies the **critical point property** (CP property) if, given any sequence satisfying $f^{k+1} \in \mathcal{A}(f^k)$, all limit points of $\{f^k\}$ are critical points of the energy.*

Analogously to the CLM property, for the SD algorithm the CP property follows as a direct consequence of lemma 1. For the IPM algorithm it follows from closedness of the minimization step.

The proof of local convergence utilizes a version of Lyapunov's direct method for set-valued maps, and we adapt this technique from the strategy outlined in [8]. We first demonstrate that if any iterate $f^k$ lies in a sufficiently small neighborhood $\mathcal{B}_\gamma(f^\infty)$ of the strict local minimum then all subsequent iterates remain in the neighborhood $\mathcal{B}_\epsilon(f^\infty)$ in which $f^\infty$ is an isolated critical point. By compactness and the CP property, any subsequence of $\{f^k\}$ must have a further subsequence that converges to the only critical point in $\mathcal{B}_\epsilon(f^\infty)$, i.e. $f^\infty$. This implies that the whole sequence must converge to $f^\infty$ as well. We formalize this argument in lemma 8 and its corollary theorem 1.

**Lemma 8** (Lyapunov Stability at Strict Local Minima). *Suppose $\mathcal{A}(f)$ is a monotonic, CSLM mapping. Fix $f^0 \in \mathcal{S}_0^{n-1}$ and let $\{f^k\}$ denote any sequence satisfying $f^{k+1} \in \mathcal{A}(f^k)$. If $f^\infty$ is a strict local minimum of the energy, then for any $\epsilon > 0$ there exists a $\gamma > 0$ so that if $f^0 \in \mathcal{B}_\gamma(f^\infty)$ then $\{f^k\} \subset \mathcal{B}_\epsilon(f^\infty)$.*

**Theorem 1** (Local Convergence at Isolated Critical Points). *Let $\mathcal{A}(f) : \mathcal{S}_0^{n-1} \rightrightarrows \mathcal{S}_0^{n-1}$ denote a monotonic, CSLM, CPP mapping. Let $f^0 \in \mathcal{S}_0^{n-1}$ and suppose $\{f^k\}$ is any sequence satisfying $f^{k+1} \in \mathcal{A}(f^k)$. Let $f^\infty$ denote a local minimum that is an isolated critical point of the energy. If $f^0 \in \mathcal{B}_\gamma(f^\infty)$ for $\gamma > 0$ sufficiently small then $f^k \to f^\infty$.*

Note that both algorithms satisfy the hypothesis of theorem 1, and therefore possess identical local convergence properties. A slight modification of the proof of theorem 1 yields the following corollary that also applies to both algorithms.

**Corollary 1.** *Let $f^0 \in \mathcal{S}_0^{n-1}$ be arbitrary, and define $f^{k+1} \in \mathcal{A}(f^k)$ according to either algorithm. If any accumulation point $f^*$ of the sequence $\{f^k\}$ is both an isolated critical point of the energy and a local minimum, then the whole sequence $f^k \to f^*$.*

# 4 Global Convergence for $\mathcal{A}_{\mathrm{SD}}$

To this point the convergence properties of both algorithms appear identical. However, we have yet to take full advantage of the superior mathematical structure afforded by the SD algorithm. In particular, from lemma 3 we know that $||f^{k+1} - f^k||_2 \to 0$ without any further assumptions regarding the initialization of the algorithm or the energy landscape. This fact combines with the fact that lemma 1 also holds globally for $f \in \mathcal{S}_0^{n-1}$ to yield theorem 2. Once again, we arrive at this conclusion by adapting the proof from [8].

**Theorem 2** (Convergence of the SD Algorithm). *Take $f^0 \in \mathcal{S}_0^{n-1}$ and fix a constant $c > 0$. Let $\{f^k\}$ denote any sequence satisfying $f^{k+1} \in \mathcal{A}_{\mathrm{SD}}(f^k)$. Then*

    *1. Any accumulation point $f^*$ of the sequence is a critical point of the energy.*

    *2. Either the sequence converges, or the set of accumulation points form a continuum in $\mathcal{S}_0^{n-1}$.*

We might hope to rule out the second possibility in statement 2 by showing that $E$ can never have an uncountable number of critical points. Unfortunately, we can exhibit (c.f. the supplementary material) simple examples to show that a continuum of local or global minima can in fact happen. This degeneracy of a continuum of critical points arises from a lack of uniqueness in the underlying combinatorial problem. We explore this aspect of convergence further in section 5.

By assuming additional structure in the energy landscape we can generalize the local convergence result, theorem 1, to yield global convergence of both algorithms. This is the content of corollary 2 for the SD algorithm and the content of corollary 3 for the IPM algorithm. The hypotheses required for each corollary clearly demonstrate the benefit of knowing apriori that $||f^{k+1} - f^k||_2 \to 0$ occurs for the SD algorithm. For the IPM algorithm, we can only deduce this a posteriori from the fact that the iterates converge.

**Corollary 2.** *Let $f^0 \in \mathcal{S}_0^{n-1}$ be arbitrary and define $f^{k+1} \in \mathcal{A}_{\mathrm{SD}}(f^k)$. If the energy has only countably many critical points in $\mathcal{S}_0^{n-1}$ then $\{f^k\}$ converges.*

**Corollary 3.** *Let $f^0 \in \mathcal{S}_0^{n-1}$ be arbitrary and define $f^{k+1} \in \mathcal{A}_{\mathrm{IPM}}(f^k)$. Suppose all critical points of the energy are isolated in $\mathcal{S}_0^{n-1}$ and are either local maxima or local minima. Then $\{f^k\}$ converges.*

While at first glance corollary 3 provides hope that global convergence holds for the IPM algorithm, our simple examples in the supplementary material demonstrate that even benign graphs with well-defined cuts have critical points of the energy that are neither local maxima nor local minima.

## 5 Energy Landscape of the Cheeger Functional

This section demonstrates that the continuous problem (2) provides an exact relaxation of the combinatorial problem (1). Specifically, we provide an explicit formula that gives an exact correspondence between the global minimizers of the continuous problem and the global minimizers of the combinatorial problem. This extends previous work [12, 11, 9] on the relationship between the global minima of (1) and (2). We also completely classifiy the local minima of the continuous problem by introducing a notion of local minimum for the combinatorial problem. Any local minimum of the combinatorial problem then determines a local minimum of the combinatorial problem by means of an explicit formula, and vice-versa. Theorem 4 provides this formula, which also gives a sharp condition for when a global minimum of the continuous problem is two-valued (binary), three-valued (trinary), or $k$-valued in the general case. This provides an understanding the energy landscape, which is essential due to the lack of convexity present in the continuous problem. Most importantly, we can classify the types of local minima encountered and when they form a continuum. This is germane to the global convergence results of the previous sections. The proofs in this section follow closely the ideas from [12, 11].

### 5.1 Local and Global Minima

We first introduce the two fundamental definitions of this section. The first definition introduces the concept of when a set $S \subset V$ of vertices is compatible with an increasing sequence $S_1 \subsetneq S_2 \subsetneq \cdots \subsetneq S_k$ of vertex subsets. Loosely speaking, a set $S$ is compatible with $S_1 \subsetneq S_2 \subsetneq \cdots \subsetneq S_k$ whenever the cut defined by the pair $(S, S^c)$ neither intersects nor crosses any of the cuts $(S_i, S_i^c)$. Definition 6 formalizes this notion.

**Definition 6** (Compatible Vertex Set). *A vertex set $S$ is **compatible** with an increasing sequence $S_1 \subsetneq S_2 \subsetneq \cdots \subsetneq S_k$ if $S \subseteq S_1$, $S_k \subseteq S$ or*

$$S_1 \subsetneq S_2 \subsetneq \cdots \subsetneq S_i \subseteq S \subseteq S_{i+1} \subsetneq \cdots \subsetneq S_k \quad \text{for some } 1 \leq i \leq k - 1,$$

The concept of compatible cuts then allows us to introduce our notion of a local minimum of the combinatorial problem, i.e. definition 7.

**Definition 7** (Combinatorial $k$-Local Minima)**.** *An increasing collection of nontrivial sets $S_1 \subsetneq S_2 \subsetneq \cdots \subsetneq S_k$ is called a **k-local minimum** of the combinatorial problem if $\mathcal{C}(S_1) = \mathcal{C}(S_2) = \cdots = \mathcal{C}(S_k) \leq \mathcal{C}(S)$ for all $S$ compatible with $S_1 \subsetneq S_2 \subsetneq \cdots \subsetneq S_k$.*

Pursuing the previous analogy, a collection of cuts $(S_1, S_1^c), \cdots, (S_k, S_k^c)$ forms a $k$-local minimum of the combinatorial problem precisely when they do not intersect, have the same energy and all other non-intersecting cuts $(S, S^c)$ have higher energy. The case of a 1-local minimum is paramount. A cut $(S_1, S_1^c)$ defines a 1-local minimum if and only if it has lower energy than all cuts that do not intersect it. As a consequence, if a 1-local minimum is not a global minimum then the cut $(S_1, S_1^c)$ necessarily intersects all of the cuts defined by the global minimizers. This is a fundamental characteristic of local minima: they are never "parallel" to global minima.

For the continuous problem, combinatorial $k$-local minima naturally correspond to vertex functions $f \in \mathbb{R}^n$ that take $(k+1)$ distinct values. We therefore define the concept of a $(k+1)$-valued local minimum of the continuous problem.

**Definition 8** (Continuous $(k+1)$-valued Local Minima)**.** *We call a vertex function $f \in \mathbb{R}^n$ a $(\mathbf{k+1})$**-valued local minimum** of the continuous problem if $f$ is a local minimum of $E$ and if its range contains exactly $k+1$ distinct values.*

Theorem 3 provides the intuitive picture connecting these two concepts of minima, and it follows as a corollary of the more technical and explicit theorem 4.

**Theorem 3.** *The continuous problem has a $(k+1)$-valued local minimum if and only if the combinatorial problem has a $k$-local minimum.*

For example, if the continuous problem has a trinary local minimum in the usual sense then the combinatorial problem must have a 2-local minimum in the sense of definition 7. As the cuts $(S_1, S_1^c)$ and $(S_2, S_2^c)$ defining a 2-local minimum do not intersect, a 2-local minimum separates the vertices of the graph into three disjoint domains. A trinary function therefore makes intuitive sense. We make this intuition precise in theorem 4. Before stating it we require two further definitions.

**Definition 9** (Characteristic Functions)**.** *Given $\emptyset \neq S \subset V$, define its **characteristic function** $f_S$ as*

$$f_S = Cut(S, S^c)^{-1} \chi_S \quad \text{if } |S| \leq n/2 \qquad \text{and} \qquad f_S = -Cut(S, S^c)^{-1} \chi_{S^c} \quad \text{if } |S| > n/2. \quad (9)$$

*Note that $f_S$ has median zero and $TV$-norm equal to 1.*

**Definition 10** (Strict Convex Hull)**.** *Given $k$ functions $f_1, \cdots, f_k$, their **strict convex hull** is the set*

$$sch\{f_1, \cdots, f_k\} = \{\theta_1 f_1 + \cdots + \theta_k f_k : \theta_i > 0 \text{ for } 1 \leq i \leq k \text{ and } \theta_1 + \cdots + \theta_k = 1\} \quad (10)$$

**Theorem 4** (Explicit Correspondence of Local Minima)**.**

1. *Suppose $S_1 \subsetneq S_2 \subsetneq \cdots \subsetneq S_k$ is a $k$-local minimum of the combinatorial problem and let $f \in sch\{f_{S_1}, \cdots, f_{S_k}\}$. Then any function of the form $g = \alpha f + \beta \mathbf{1}$ defines a $(k+1)$-valued local minimum of the continuous problem and with $E(g) = \mathcal{C}(S_1)$.*

2. *Suppose that $f$ is a $(k+1)$-valued local minimum and let $c_1 > c_2 > \cdots > c_{k+1}$ denote its range. For $1 \leq i \leq k$ set $\Omega_i = \{f = c_i\}$. Then the increasing collection of sets $S_1 \subsetneq \cdots \subsetneq S_k$ given by*

$$S_1 = \Omega_1, \quad S_2 = \Omega_1 \cup \Omega_2 \qquad \cdots \qquad S_k = \Omega_1 \cup \cdots \cup \Omega_k$$

   *is a $k$-local minimum of the combinatorial problem with $\mathcal{C}(S_i) = E(f)$.*

**Remark 1** (Isolated vs Continuum of Local Minima)**.** *If a set $S_1$ is a 1-local min then the strict convext hull (10) of its characteristic function reduces to the single binary function $f_{S_1}$. Thus every 1-local minimum generates exactly one local minimum of the continuous problem in $\mathcal{S}_0^{n-1}$, and this local minimum is binary. On the other hand, if $k \geq 2$ then every $k$-local minimum of the combinatorial problem generates a continuum (in $\mathcal{S}_0^{n-1}$) of non-binary local minima of the continuous problem. Thus, the hypotheses of theorem 1, corollary 2 or corollary 3 can hold only if no such higher order $k$-local minima exist. When these theorems do apply the algorithms therefore converge to a binary function.*

As a final consequence, we summarize the fact that theorem 4 implies that the continuous relaxation of the Cheeger cut problem is exact. In other words,

**Theorem 5.** *Given* $\{f \in \arg\min E\}$ *an explicit formula exists to construct the set* $\{S \in \arg\min \mathcal{C}\}$, *and vice-versa.*

## 6 Experiments

In all experiments, we take the constant $c = 1$ in the SD algorithm. We use the method from [3] to solve the minimization problem in the SD algorithm and the method from [7] to solve the minimization problem in the IPM algorithm. We terminate each minimization when either a stopping tolerance of $\varepsilon = 10^{-10}$ (i.e. $\|u^{j+1} - u^j\|_1 \leq \varepsilon$) or $2,000$ iterations is reached. This yields a comparison of the idealized cases of the SD algorithm and the IPM algorithm. Our first experiment uses the two-moon dataset [2] in the same setting as in [12]. The second experiment utilizes pairs of image digits extracted from the MNIST dataset. The first table summarizes the results of these tests. It shows the mean Cheeger energy value (2), the mean error of classification (% of misclassified data) and the mean computational time for both algorithms over $10$ experiments with the same random initialization for both algorithms in each of the individual experiments.

| | SD Algorithm | | | Modified IPM Algorithm [7] | | |
|---|---|---|---|---|---|---|
| | Energy | Error (%) | Time (sec.) | Energy | Error (%) | Time (sec.) |
| 2 moons | 0.126 | 8.69 | 2.06 | 0.145 | 14.12 | 1.98 |
| 4's and 9's | 0.115 | 1.65 | 52.4 | 0.185 | 25.23 | 58.9 |
| 3's and 8's | 0.086 | 1.217 | 49.2 | 0.086 | 1.219 | 48.1 |

Our second set of experiments applies both algorithms to multi-class clustering problems using a standard, recursive bi-partitioning method. We use the MNIST, USPS and COIL datasets. We preprocessed the data by projecting onto the first 50 principal components, and take $k = 10$ nearest neighbors for the MNIST and USPS datasets and $k = 5$ nearest neighbors for the COIL dataset. We used the same tolerances for the minimization problems, i.e. $\varepsilon = 10^{-10}$ and $2,000$ maximum iterations. The table below presents the mean Cheeger energy, classification error and time over $10$ experiments as before.

| | SD Algorithm | | | Modified IPM Algorithm [7] | | |
|---|---|---|---|---|---|---|
| | Energy | Err. (%) | Time (min.) | Energy | Err. (%) | Time (min.) |
| MNIST (10 classes) | 1.30 | 11.78 | 45.01 | 1.29 | 11.75 | 42.83 |
| USPS (10 classes) | 2.37 | 4.11 | 5.15 | 2.37 | 4.13 | 4.81 |
| COIL (20 classes) | 0.19 | 1.58 | 4.31 | 0.18 | 2.52 | 4.20 |

Overall, the results show that both algorithms perform equivalently for both two-class and multi-class clustering problems.

As our interest here lies in the theoretical properties of both algorithms, we will study practical implementation details for the SD algorithm in future work. For instance, as Hein and Bühler remark [6], solving the minimization problem for the IPM algorithm precisely is unnecessary. Analogously for the SD Algorithm, we only need to lower the energy sufficiently before proceeding to the next iteration of the algorithm. It proves convenient to stop the minimization when a weaker form of the energy inequality (6) holds, such as

$$E(f) \geq E(h) + \theta \left( \frac{E(f)}{B(h)} \frac{\|\hat{h} - f\|_2^2}{c} \right)$$

for some constant $0 < \theta < 1$. This condition provably holds in a finite number of iterations and still guarantees that $\|f^{k+1} - f^k\|_2 \to 0$. The concrete decay estimate provided by SD algorithm therefore allows us to give precise meaning to "sufficiently lowers the energy." We investigate these aspects of the algorithm and prove convergence for this practical implementation in future work.

**Reproducible research:** The code is available at http://www.cs.cityu.edu.hk/~xbresson/codes.html

**Acknowledgements:** This work supported by AFOSR MURI grant FA9550-10-1-0569 and Hong Kong GRF grant #110311.

# References

[1] X. Bresson, X.-C. Tai, T.F. Chan, and A. Szlam. Multi-Class Transductive Learning based on $\ell^1$ Relaxations of Cheeger Cut and Mumford-Shah-Potts Model. *UCLA CAM Report*, 2012.

[2] T. Bühler and M. Hein. Spectral Clustering Based on the Graph p-Laplacian. In *International Conference on Machine Learning*, pages 81–88, 2009.

[3] A. Chambolle and T. Pock. A First-Order Primal-Dual Algorithm for Convex Problems with Applications to Imaging. *Journal of Mathematical Imaging and Vision*, 40(1):120–145, 2011.

[4] J. Cheeger. A Lower Bound for the Smallest Eigenvalue of the Laplacian. *Problems in Analysis*, pages 195–199, 1970.

[5] F. R. K. Chung. *Spectral Graph Theory*, volume 92 of *CBMS Regional Conference Series in Mathematics*. Published for the Conference Board of the Mathematical Sciences, Washington, DC, 1997.

[6] M. Hein and T. Bühler. An Inverse Power Method for Nonlinear Eigenproblems with Applications in 1-Spectral Clustering and Sparse PCA. In *In Advances in Neural Information Processing Systems (NIPS)*, pages 847–855, 2010.

[7] M. Hein and S. Setzer. Beyond Spectral Clustering - Tight Relaxations of Balanced Graph Cuts. In *In Advances in Neural Information Processing Systems (NIPS)*, 2011.

[8] R.R. Meyer. Sufficient conditions for the convergence of monotonic mathematical programming algorithms. *Journal of Computer and System Sciences*, 12(1):108 – 121, 1976.

[9] S. Rangapuram and M. Hein. Constrained 1-Spectral Clustering. In *International conference on Artificial Intelligence and Statistics (AISTATS)*, pages 1143–1151, 2012.

[10] J. Shi and J. Malik. Normalized Cuts and Image Segmentation. *IEEE Transactions on Pattern Analysis and Machine Intelligence (PAMI)*, 22(8):888–905, 2000.

[11] G. Strang. Maximal Flow Through A Domain. *Mathematical Programming*, 26:123–143, 1983.

[12] A. Szlam and X. Bresson. Total variation and cheeger cuts. In *Proceedings of the 27th International Conference on Machine Learning*, pages 1039–1046, 2010.

[13] L. Zelnik-Manor and P. Perona. Self-tuning Spectral Clustering. In *In Advances in Neural Information Processing Systems (NIPS)*, 2004.

